# An Orientation Selective Neural Network for Pattern Identification in Particle Detectors

**Halina Abramowicz, David Horn, Ury Naftaly, Carmit Sahar–Pikielny**
School of Physics and Astronomy, Tel Aviv University
Tel Aviv 69978, Israel
halina@post.tau.ac.il, horn@neuron.tau.ac.il
ury@post.tau.ac.il, carmit@post.tau.ac.il

## Abstract

We present an algorithm for identifying linear patterns on a two-dimensional lattice based on the concept of an orientation selective cell, a concept borrowed from neurobiology of vision. Constructing a multi-layered neural network with fixed architecture which implements orientation selectivity, we define output elements corresponding to different orientations, which allow us to make a selection decision. The algorithm takes into account the granularity of the lattice as well as the presence of noise and inefficiencies. The method is applied to a sample of data collected with the ZEUS detector at HERA in order to identify cosmic muons that leave a linear pattern of signals in the segmented calorimeter. A two dimensional representation of the relevant part of the detector is used. The algorithm performs very well. Given its architecture, this system becomes a good candidate for fast pattern recognition in parallel processing devices.

## I Introduction

A typical problem in experiments performed at high energy accelerators aimed at studying novel effects in the field of Elementary Particle Physics is that of preselecting interesting interactions at as early a stage as possible, in order to keep the data volume manageable. One class of events that have to be eliminated is due to cosmic muons that pass all trigger conditions.

The most characteristic feature of cosmic muons is that they leave in the detector a path of signals aligned along a straight line. The efficiency of pattern recognition algorithms depends strongly on the granularity with which such a line is probed, on the level of noise and the response efficiency of a given detector. Yet the efficiency of a visual scan is fairly independent of those features [1] . This lead us to look for a new approach through application of ideas from the field of vision.

The main tool that we borrow from the neuronal circuitry of the visual cortex is the orientation selective simple cell [2]. It is incorporated in the hidden layers of a feed forward neural network, possessing a predefined receptive field with excitatory and inhibitory connections. Using these elements we have developed [3] a method for identifying straight lines of varying slopes and lengths on a grid with limited resolution. This method is then applied to the problem of identifying cosmic muons in accelerator data, and compared with other tools.

By using a network with a fixed architecture we deviate from conventional approaches of neural networks in particle physics [4]. One advantage of this approach is that the number of free parameters is small, and it can, therefore, be determined using a small data set. The second advantage is the fact that it opens up the possibility of a relatively simple implementation in hardware. This is an important feature for particle detectors, since high energy physics experiments are expected to produce in the next decade a flux of data that is higher than present analysis methods can cope with.

## II   Description of the Task

In a two-dimensional representation, the granularity of the rear part of the ZEUS calorimeter [6] can be emulated roughly by a $23 \times 23$ lattice of $20 \times 20$ cm$^2$ squares. While such a representation does not use the full information available in the detector, it is sufficient for our study. In our language each cell of this lattice will be denoted as a pixel. A pixel is activated if the corresponding calorimeter cell is above a threshold level predetermined by the properties of the detector.

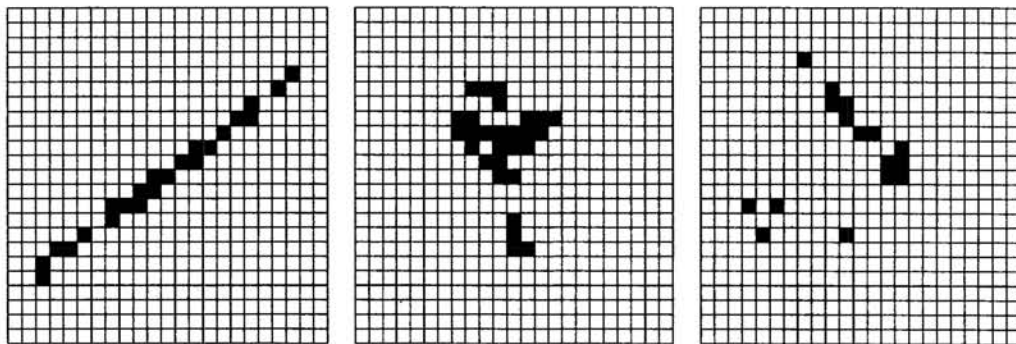

Figure 1: Example of patterns corresponding to a cosmic muon (left), a typical accelerator event (middle), and an accelerator event that looks like a muon (right), as seen in a two dimensional projection.

A cosmic muon, depending on its angle of incidence, activates along its linear path typically from 3 to 25 neighboring pixels anywhere on the $23 \times 23$ grid. The pattern of signals generated by accelerator events consists on average of 3 to 8 clusters, of typically 4 adjacent activated pixels, separated by empty pixels. The clusters

tend to populate the center of the $23 \times 23$ lattice. Due to inherent dynamics of the interactions under study, the distribution of clusters is not isotropic. Examples of events, as seen in the two-dimensional projection in the rear part of the ZEUS calorimeter, are shown in figure 1.

The lattice discretizes the data and distorts it. Adding conventional noise levels, the decision of classification of the data into accelerator events and cosmic muon events is difficult to obtain through automatic means. Yet, it is the feeling of experimentalists dealing with these problems, that any expert can distinguish between the two cases with high efficiency (identifying a muon as such) and purity (not misidentifying an accelerator event). We define our task as developing automatic means of doing the same.

## III   The Orientation Selective Neural Network

Our analysis is based on a network of orientation selective neurons (OSNN) that will be described in this chapter. We start out with an input layer of pixels on a two dimensional grid with discrete labeling $i = (x, y)$ of the neuron (pixel) that can get the values $S_i = 1$ or 0, depending on whether the pixel is activated or not.

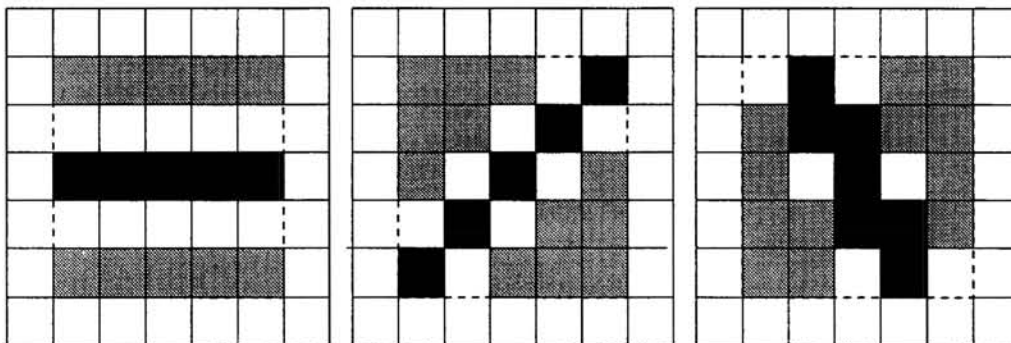

Figure 2: Connectivity patterns for orientation selective cells on the second layer of the OSNN. From left to right are examples of orientations of 0, $\pi/4$ and $5\pi/8$. Non-zero weights are defined only within a $5 \times 5$ grid. The dark pixels have weights of +1, and the grey ones have weights of -1. White pixels have zero weights.

The input is being fed into a second layer that is composed of orientation selective neurons $V_2^{i,\alpha}$ at location $i$ with orientation $\theta_\alpha$ where $\alpha$ belongs to a discrete set of 16 labels, i.e. $\theta_\alpha = \alpha\pi/16$. The neuron $V_2^{i,\alpha}$ is the analog of a simple cell in the visual cortex. Its receptive field consists of an array of dimension $5 \times 5$ centered at pixel $i$. Examples of the connectivity, for three different choices of $\alpha$, are shown in Fig. 2. The weights take the values of 1,0 and -1.

The second layer consists then of $23 \times 23 \times 16$ neurons, each of which may be thought of as one of 16 orientation elements at some $(x, y)$ location of the input layer. Next we employ a modified Winner Take All (WTA) algorithm, selecting the leading orientation $\alpha_{max}(i)$ for which the largest $V_2^{i,\alpha}$ is obtained at the given location $i$. If we find that several $V_2^{i,\alpha}$ at the same location $i$ are close in value to the maximal one, we allow up to five different $V_2^{i,\alpha}$ neurons to remain active at this stage of the processing, provided they all lie within a sector of $\alpha_{max} \pm 2$, or $\theta_{max} \pm \pi/8$. All other $V_2^{i,\alpha}$ are reset to zero. If, however, at a given location $i$ we obtain several

large values of $V_2^{i,\alpha}$ that correspond to non-neighboring orientations, all are being discarded.

The third layer also consists of orientation selective cells. They are constructed with a receptive field of size $7 \times 7$, and receive inputs from neurons with the same orientation on the second layer. The weights on this layer are defined in a similar fashion to the previous ones, but here negative weights are assigned the value of $-3$, not $-1$. For linear patterns, the purpose of this layer is to fill in the holes due to fluctuations in the pixel activation, i.e. complete the lines of same orientation of the second layer. As before, we keep also here up to five highest values at each location, following the same WTA procedure as on the second layer.

The fourth layer of the OSNN consists of only 16 components, $D^\alpha$, each corresponding to one of the discrete orientations $\alpha$. For each orientation we calculate the convolution of the first and third layers, $D^\alpha = \sum_i V_3^{i,\alpha} S_i$ . The elements $D^\alpha$ carry the information about the number of the input pixels that contribute to a given orientation $\theta_\alpha$. Cosmic muons are characterized by high values of $D^\alpha$ whereas accelerator events possess low values, as shown in figure 3 below.

The computational complexity of this algorithm is $\mathcal{O}(n)$ where $n$ is the number of pixels, since a constant number of operations is performed on each pixel. There are basically four free parameters in the algorithm. These are the sizes of the receptive fields on the second and third layer and the corresponding activation thresholds. Their values can be tuned for the best performance, however they are well constrained by the spatial resolution, the noise level in the system and the activation properties of the input pixels. The size of the receptive field determines to a large extent the number of orientations allowed to survive in the modified WTA algorithm.

## IV    OSNN and a Selection Criterion on the Training Set

The details of the design of the OSNN and the tuning of its parameters were fixed while training it on a sample of 250 cosmic muons and a similar amount of accelerator events. The sample was obtained by preselection with existing algorithms and a visual scan as a cross-check.

For cosmic muon events the highest value of $D^\alpha$, $D_{\max}$, determines the orientation of the straight line. In figure 3 we present the correlation between $D_{\max}$ and the number $n_p$ of activated input pixels for cosmic muon and accelerator events. As expected one observes a linear correlation between $D_{\max}$ and $n_p$ for the muons while almost no correlation is observed for accelerator events. This allows us to set a selection criterion defined by the separator in this figure. We quantify the quality of our selection by quoting the efficiency of properly identifying a cosmic muon for 100% purity, corresponding to no accelerator event misidentified as a muon. In OSNN-$D$, which we define according to the separator shown in Fig 3, we obtain 93.0% efficiency on the training set.

On the right hand side of Fig 3 we present results of a conventional method for detecting lines on a grid, the Hough transform [7, 8, 9]. This is based on the analysis of a parameter space describing locations and slopes of straight lines. The cells of this space with the largest occupation number, $N_{\max}$, are the analogs of our $D_{\max}$. In the figure we show the correlation of $N_{\max}$ with $n_p$ which allows us to draw a separator between cosmic muons and accelerator events, leading to an efficiency of 88% for 100% purity. Although this number is not much lower than the

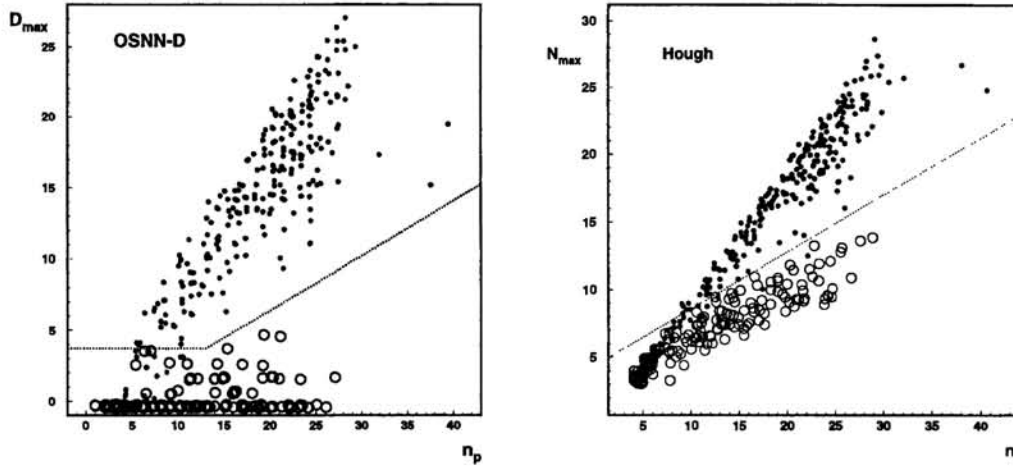

Figure 3: Left: Correlation between the maximum value of $D^\alpha$, $D_{\max}$, and the number $n_p$ of input pixels for cosmic muon (dots) and accelerator events (open circles). The dashed line defines a separator such that all events above it correspond to cosmic muons (100% purity). This selection criterion has 93% efficiency. Right: Using the Hough Transform method, we compare the values of the largest accumulation cell $N_{\max}$ with $n_p$ and find that the two types of events have different slopes, thus allowing also the definition of a separator. In this case, the efficiency is 88%.

efficiency of OSNN-$D$, we note that the difference between the two types of event distributions is not as significant as in OSNN-$D$. In the test set, to be discussed in the next chapter, we will consider 40,000 accelerator events contaminated by less than 100 cosmic muons. Clearly the expected generalization quality of OSNN-$D$ will be higher than that of the Hough transform. It should of course be noted that the OSNN is a multi-layer network, whereas the Hough transform method that we have described is a single-layer operation, i.e. it calculates global characteristics. If one wishes to employ some quasi-local Hough transform one is naturally led back to a network that has to resemble our OSNN.

## V   Training and Testing of OSNN-$S$

If instead of applying a simple cut we employ an auxiliary neural network to search for the best classification of events using the OSNN outputs, we obtain still better results. The auxiliary network has 6 inputs, one hidden layer with 5 nodes and one output unit. The input consists of a set of five consecutive $D^\alpha$ centered around $D_{\max}$ and the total number of activated input pixels, $n_p$. The cosmic muons are assigned an output value $s = 1$ and the accelerator events $s = 0$. The net is trained on our sample with error back-propagation. This results in an improved separation of cosmic muon events from the rest. Whereas in OSNN-$D$ we find a continuum of cosmic muons throughout the range of $D_{\max}$, here we obtain a clear bimodal distribution, as seen in Figure 4. For $s \geq 0.1$ no accelerator events are found and the muons are selected with an efficiency of 94.7%. This selection procedure will be denoted as OSNN-$S$.

As a test of our method we apply OSNN-$S$ to a sample of 38,606 data events

that passed the standard physics cuts [5]. The distribution of the neural network output $s$ is presented in Figure 4. It looks very different from the one obtained with the training sample. Whereas the former consisted of approximately 500 events distributed equally among accelerator events and cosmic muons, this one contains mostly accelerator events, with a fraction of a percent of muons. This proportion is characteristic of physics samples. The vast majority of accelerator events are found in the first bin, but a long tail extends throughout $s$. The last bin in $s$ is indeed dominated by cosmic muons.

We performed a visual scan of all 181 events with $s \geq 0.1$ using the full information from the detector. This allowed us to identify the cosmic-muon events represented by shaded areas in figure 4. For $s \geq 0.1$ we find 55 cosmic-muon events and 123 accelerator events, 55 of which resemble muons on the rear segment of the calorimeter. The latter, together with the genuine cosmic muons, populate mainly the region of large $s$ values.

We conclude that our method picked out the cosmic muons from the very large sample of data, in spite of the fact that it relied just on two-dimensional information from the rear part of the detector. This fact is, however, responsible for the contamination of the high $s$ region by accelerator events that resemble cosmic muons. Even with all its limitations, our method reduces the problem of rejecting cosmic-muon events down to scanning less than one percent of all the events. We conclude that we have achieved the goal that we set for ourselves, that of replacing a laborious visual scan by a computer algorithm with similar reliability.

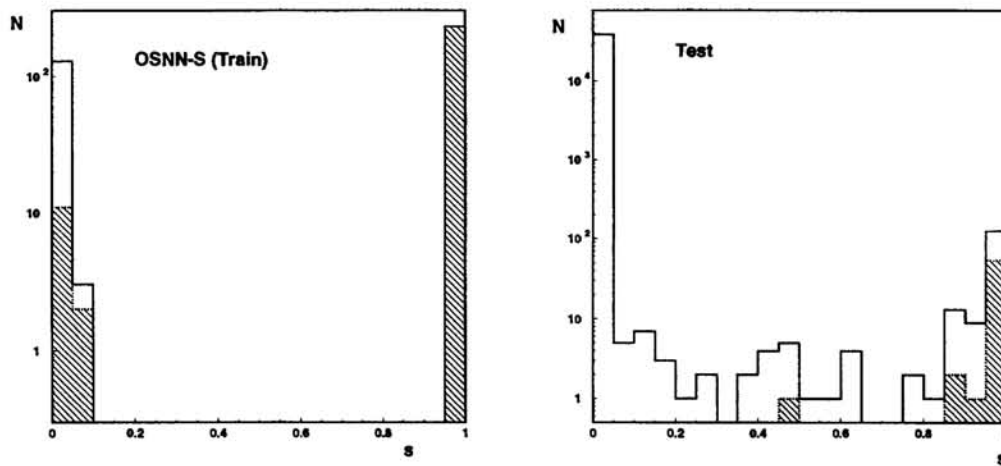

Figure 4: Left: Number of events as a function of the output $s$ of an auxiliary neural net. Choosing the separator to be $s = 0.1$ we obtain an efficiency of 94.7% on our training set. This bimodal distribution holds the promise of better generalization than the OSNN-$D$ method depicted in Figure 3. Muons are represented by shaded areas. Right: Distribution of the auxiliary neural network output $s$ obtained with the OSNN-$S$ selector for the test sample of 38,606 events. The tail of the distribution of accelerator events leads to 123 accelerator events with $s > 0.1$, including 55 that resemble straight lines on the input layer. 55 genuine cosmic muons were identified in the high $s$ region.

## VI  Summary

We have presented an algorithm for identifying linear patterns on a two-dimensional lattice based on the concept of an orientation selective cell, a concept borrowed from neurobiology of vision. Constructing a multi-layered neural network with fixed architecture that implements orientation selectivity, we define output elements corresponding to different orientations, that allow us to make a selection decision. The algorithm takes into account the granularity of the lattice as well as the presence of noise and inefficiencies.

Our feed-forward network has a fixed set of synaptic weights. Hence, although the number of neurons is very high, the complexity of the system, as determined by the number of free parameters, is low. This allows us to train our system on a small data set. We are gratified to see that, nontheless, it generalizes well and performs excellently on a test sample that is larger by two orders of magnitude.

One may regard our method as a refinement of the Hough transform, since each of our orientation selective cells acts as a filter of straight lines on a limited grid. The major difference from conventional Hough transforms is that we perform semi-local calculations, and proceed in several stages, reflected by the different layers of our network, before evaluating global parameters.

The task that we have set to ourselves in the application described here is only one example of problems of pattern recognition that are encountered in the analysis of particle detectors. Given the large flux of data in these experiments, one is faced by two requirements: correct identification and fast performance. Using a structure like our OSNN for data classification, one can naturally meet the speed requirement through its realization in hardware, taking advantage of the basic features of distributed parallel computation.

### Acknowledgements

We are indebted to the ZEUS Collaboration for allowing us to use the sample of data for this analysis. This work was partly supported by a grant from the Israel Science Foundation.

## References

[1] ZEUS Collab., The ZEUS Detector, Status Report 1993, DESY 1993; M. Derrick et al., Phys. Lett. B 293 (1992) 465.

[2] D. H. Hubel and T. N. Wiesel, J. Physiol. 195 (1968) 215.

[3] H. Abramowicz, D. Horn, U. Naftaly and C. Sahar-Pikielny, Nuclear Instrum. and Methods in Phys. Res. A378 (1996) 305.

[4] B. Denby, Neural Computation, 5 (1993) 505.

[5] ZEUS Calorimeter Group, A. Andresen et al., Nucl. Inst. Meth. A 309 (1991) 101.

[6] P. V. Hough, "Methods and means to recognize complex patterns", U.S. patent 3.069.654.

[7] D. H. Ballard, Pattern Recognition 3 (1981) 11.

[8] R. O. Duda and P. E. Hart, Commun. ACM. 15 (1972) 1.

[9] ZEUS collab., M. Derrick et al., Phys. Lett. B 316 (1993) 412; ZEUS collab., M. Derrick et al., Zeitschrift f. Physik C 69 (1996) 607-620